# GRIFT: A graphical model for inferring visual classification features from human data

**Michael G. Ross**
Department of Brain and Cognitive Sciences
Massachusetts Institute of Technology
Cambridge, MA 02139
mgross@mit.edu

**Andrew L. Cohen**
Psychology Department
University of Massachusetts Amherst
Amherst, MA 01003
acohen@psych.umass.edu

## Abstract

This paper describes a new model for human visual classification that enables the recovery of image features that explain human subjects' performance on different visual classification tasks. Unlike previous methods, this algorithm does not model their performance with a single linear classifier operating on raw image pixels. Instead, it represents classification as the combination of multiple feature detectors. This approach extracts more information about human visual classification than previous methods and provides a foundation for further exploration.

## 1 Introduction

Although a great deal is known about the low-level features computed by the human visual system, determining the information used to make high-level visual classifications is an active area of research. When a person distinguishes between two faces, for example, what image regions are most salient? Since the early 1970s, one of the most important research tools for answering such questions has been the *classification image* (or *reverse correlation*) algorithm, which assumes a linear classification model [1]. This paper describes a new approach, GRIFT (GRaphical models for Inferring Feature Templates). Instead of representing human visual discrimination as a single linear classifier, GRIFT models it as the non-linear combination of multiple independently detected features. This allows GRIFT to extract more detailed information about human classification.

This paper describes GRIFT and the algorithms for fitting it to data, demonstrates the model's efficacy on simulated and human data, and concludes with a discussion of future research directions.

## 2 Related work

Ahumada's classification image algorithm [1] models an observer's classifications of visual stimuli with a noisy linear classifier — a fixed set of weights and a normally distributed threshold. The random threshold accounts for the fact that multiple presentations of the same stimulus are often classified inconsistently. In a typical classification image experiment, participants are presented with hundreds or thousands of noise-corrupted examples from two categories and asked to classify each one. The noise ensures that the samples cover a large volume of the sample space in order to allow recovery of a unique linear classifier that best explains the data.

Although classification images are useful in many cases, it is well established that there are domains in which recognition and classification are the result of combining the detection of parts or features, rather than applying a single linear template. For example, Pelli et al. [10], have convincingly demonstrated that humans recognize noisy word images by parts, even when whole-word templates would perform better. Similarly, Gold et al. [7] verified that subjects employed feature-based clas-

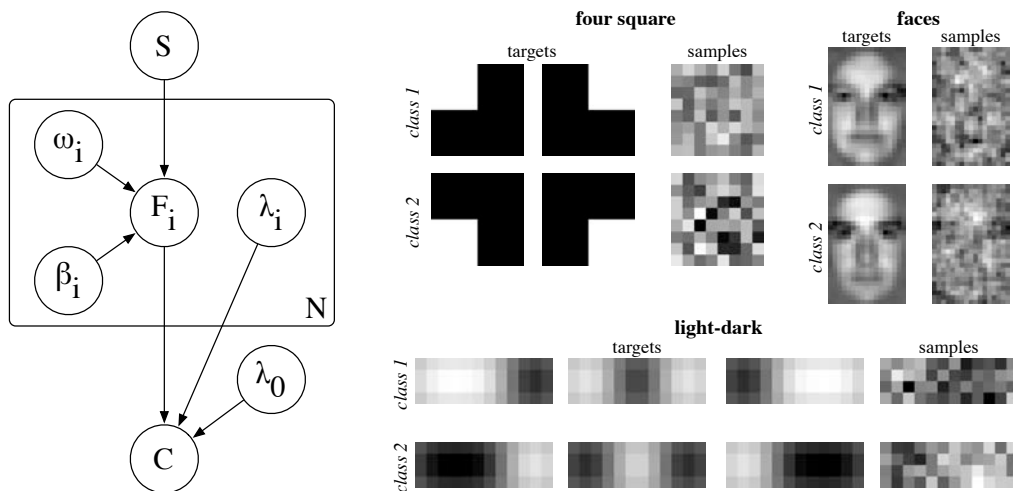

Figure 1: Left: The GRIFT model is a Bayes net that describes classification as the result of combining $N$ feature detectors. Right: Targets and sample stimuli from the three experiments.

sification strategies for some simple artificial image classes. GRIFT takes the next step and infers features which predict human performance directly from classification data.

Most work on modeling non-linear, feature-based classification in humans has focused on verifying the use of a predefined set of features. Recent work by Cohen et al. [4] demonstrates that Gaussian mixture models can be used to recover features from human classification data without specifying a fixed set of possible features. The GRIFT model, described in the remainder of this paper, has the same goals as the previous work, but removes several limitations of the Gaussian mixture model approach, including the need to only use stimuli the subjects classified with high confidence and the bias that the signals can exert on the recovered features. GRIFT achieves these and other improvements by generatively modeling the entire classification process with a graphical model. Furthermore, the similarity between single-feature GRIFT models and the classification image process, described in more detail below, make GRIFT a natural successor to the traditional approach.

## 3   GRIFT model

GRIFT models classification as the result of combining $N$ conditionally independent feature detectors, $F = \{F_1, F_2, \ldots, F_N\}$. Each feature detector is binary valued (1 indicates detection), as is the classification, $C$ (1 indicates one class and 2 the other). The stimulus, $S$, is an array of continuously valued pixels representing the input image. The stimulus only influences $C$ through the feature detectors, therefore the joint probability of a stimulus and classification pair is

$$P(C,S) = \sum_F \left( P(C|F)P(S) \prod_i^N P(F_i|S) \right).$$

Figure 1 represents the causal relationship between these variables ($C$, $F$, and $S$) with a Bayesian network. The network also includes nodes representing model parameters ($\omega$, $\beta$, and $\lambda$), whose role will be described below. The boxed region in the figure indicates the parts of the model that are replicated when $N > 1$ — each feature detector is represented by an independent copy of those variables and parameters.

The distribution of the stimulus, $P(S)$, is under the control of the experimenter. The algorithm for fitting the model to data only assumes that the stimuli are independent and identically distributed across trials. The conditional distribution of each feature detector's value, $P(F_i|S)$, is modeled with a logistic regression function on the pixel values of $S$. Logistic regression is desirable because it is a probabilistic linear classifier. Humans can successfully classify images in the presence of extremely high additive noise, which suggests the use of averaging and contrast, linear computations which

are known to play important roles in human visual perception [9]. Just as the classification image used a random threshold to represent uncertainty in the output of its single linear classifier, logistic regression also allows GRIFT to represent uncertainty in the output of each of its feature detectors. The conditional distribution of $C$ is represented by logistic regression on the feature outputs.

Each $F_i$'s distribution has two parameters, a weight vector $\omega_i$ and a threshold $\beta_i$, such that

$$P(F_i = 1|S, \omega_i, \beta_i) = (1 + \exp(\beta_i + \sum_{j=1}^{|S|} \omega_{ij} S_j))^{-1},$$

where $|S|$ is the number of pixels in a stimulus. Similarly, the conditional distribution of $C$ is determined by $\lambda = \{\lambda_0, \lambda_1, \ldots, \lambda_N\}$ where

$$P(C = 1|F, \lambda) = (1 + \exp(\lambda_0 + \sum_{i=1}^{N} \lambda_i F_i))^{-1}.$$

Detecting a feature with negative $\lambda_i$ increases the probability that the subject will respond "class 1," those with positive $\lambda_i$ are associated with "class 2" responses.

A GRIFT model with $N$ features applied to the classification of images each containing $|S|$ pixels has $N(|S| + 2) + 1$ parameters. This large number of parameters, coupled with the fact that the F variables are unobservable, makes fitting the model to data very challenging. Therefore, GRIFT defines prior distributions on its parameters. These priors reflect reasonable assumptions about the parameter values and, if they are wrong, can be overturned if enough contrary data is available. The prior on each of the $\lambda_i$ parameters for which $i > 0$ is a mixture of two normal distributions,

$$P(\lambda_i) = \frac{1}{2\sqrt{2\pi}}(\exp(-\frac{(\lambda_i + 2)^2}{2}) + \exp(-\frac{(\lambda_i - 2)^2}{2})).$$

This prior reflects the assumption that each feature detector should have a significant impact on the classification, but no single detector should make it deterministic — a single-feature model with $\lambda_0 = 0$ and $\lambda_1 = -2$ has an 88% chance of choosing class 1 if the feature is active. The $\lambda_0$ parameter has an improper non-informative prior, $P(\lambda_0) = 1$, indicating no preference for any particular value [5] because the best $\lambda_0$ is largely determined by the other $\lambda_i$s and the distributions of $F$ and $S$. For analogous reasons, $P(\beta_i) = 1$.

The $\omega_i$ parameters, which each have dimensionality equal to the stimulus, present the biggest inferential challenge. As mentioned previously, human visual processing is sensitive to contrasts between image regions. If one image region is assigned positive $\omega_{ij}$s and another is assigned negative $\omega_{ij}$s, the feature detector will be sensitive to the contrast between them. This contrast between regions requires all the pixels within each region to share similar $\omega_{ij}$ values. To encourage this local structure, the $\omega_i$ parameters have Markov random field prior distributions:

$$P(\omega_i) \propto \left( \prod_j (\exp(-\frac{(\omega_{ij} + 1)^2}{2}) + \exp(-\frac{(\omega_{ij} - 1)^2}{2})) \right) \left( \prod_{(j,k) \in A} \exp(-\frac{(\omega_{ij} - \omega_{ik})^2}{2}) \right),$$

where $A$ is the set of neighboring pixel locations. The first factor encourages weight values to be near the -1 to 1 range, while the second encourages the assignment of similar weights to neighboring pixels. Fitting the model to data does not require the normalization of this distribution.

The Bayesian joint probability distribution of all the parameters and variables is

$$P(C, F, S, \omega, \beta, \lambda) = P(C|F, \lambda)P(S)P(\lambda_0) \prod_{i=1}^{N} P(F_i|S, \omega_i, \beta_i)P(\omega_i)P(\beta_i)P(\lambda_i). \quad (1)$$

## 4 GRIFT algorithm

The goal of the algorithm is to find the parameters that satisfy the prior distributions and best account for the $(S, C)$ samples gathered from a human subject. Mathematically, this goal corresponds to finding the mode of $P(\omega, \beta, \lambda|\mathbf{S}, \mathbf{C})$, where $\mathbf{S}$ and $\mathbf{C}$ refer to all of the observed samples. The

algorithm is derived using the expectation-maximization (EM) method [3], a widely used optimization technique for dealing with unobserved variables, in this case $\mathbf{F}$, the feature detector outputs for all the trials. In order to determine the most probable parameter assignments, the algorithm chooses random initial parameters $\theta^* = (\omega^*, \beta^*, \lambda^*)$ and then finds the $\theta$ that maximizes

$$Q(\theta|\theta^*) = \sum_{\mathbf{F}} P(\mathbf{F}|\mathbf{S}, \mathbf{C}, \theta^*) \log P(\mathbf{C}, \mathbf{F}, \mathbf{S}|\theta) + \log P(\theta).$$

$Q(\theta|\theta^*)$ is the expected log posterior probability of the parameters computed by using the current $\theta^*$ to estimate the distribution of $\mathbf{F}$, the unobserved feature detector activations. The $\theta$ that maximizes $Q$ then becomes $\theta^*$ for the next iteration, and the process is repeated until convergence.

The presence of both the $P(\mathbf{C}, \mathbf{F}, \mathbf{S}|\theta)$ and $P(\theta)$ terms encourages the algorithm to find parameters that explain the data and match the assumptions encoded in the parameter prior distributions. As the amount of available data increases, the influence of the priors decreases, so it is possible to discover features that are contrary to prior belief given enough evidence.

Using the conditional independences from the Bayes net:

$$\begin{aligned}
Q(\theta|\theta^*) \quad &\propto \quad \sum_{\mathbf{F}} P(\mathbf{F}|\mathbf{S}, \mathbf{C}, \theta^*) \left( \log P(\mathbf{C}|\mathbf{F}, \lambda) + \sum_{i=1}^{N} \log P(\mathbf{F_i}|\mathbf{S}, \omega_i, \beta_i) \right) \\
&+ \sum_{i=1}^{N} \left( \log P(\omega_i) + \log P(\lambda_i) \right),
\end{aligned}$$

dropping the $\log P(\mathbf{S})$ term, which is independent of the parameters, and the $\log P(\lambda_0)$ and $\log P(\beta_i)$ terms, which are 0. As mentioned before, the normalization terms for the $\log P(\omega_i)$ elements can be ignored during optimization — the $\log$ makes them additive constants to $Q$. The functional form of every additive term is described in Section 3, and $P(\mathbf{F}|\mathbf{S}, \mathbf{C}, \theta^*)$ can be calculated using the model's joint probability function (Equation 1).

Each iteration of EM requires maximizing $Q$, but it is not possible to compute the maximizing $\theta$ in closed form. Fortunately, it is relatively easy to search for the best $\theta$. Because $Q$ is separable into many additive components, it is possible to efficiently compute its gradient with respect to each of the elements of $\theta$ and use this information to find a locally maximum $\theta$ assignment using the scaled conjugate gradient algorithm [2]. Even a locally maximum value of $\theta$ usually provides good EM results — $P(\omega, \beta, \lambda|\mathbf{S}, \mathbf{C})$ is still guaranteed to improve after every iteration.

The result of any EM procedure is only guaranteed to be a locally optimal answer, and finding the globally optimal $\theta$ is made more challenging by the large number of parameters. GRIFT adopts the standard solution of running EM many times, each instance starting with a random $\theta^*$, and then accepting the $\theta$ from the run which produced the most probable parameters. For this model and the data presented in the following sections, 20-30 random restarts were sufficient.

## 5   Experiments

The GRIFT model was fit to data from 3 experiments. In each experiment, human participants classified stimuli into two classes. Each class contained one or more target stimuli. In each trial, the participant saw a stimulus (a sample from $S$) that consisted of a randomly chosen target with high levels of independent identically distributed noise added to each pixel. The noise samples were drawn from a truncated normal distribution to ensure that the stimulus pixel values remained within the display's output range. Figure 1 shows the classes and targets from each experiment and a sample stimulus from each class. In the *four-square* experiment four participants were asked to distinguish between two artificial stimulus classes, one in which there were bright squares in the upper-left or upper-right corners and one in which there were bright squares in the lower-left or lower-right corners. In the *light-dark* experiment three participants were asked to distinguish between three strips that each had two light blobs and three strips that each had only one light blob. Finally, in the *faces* experiment three participants were asked to distinguish between two faces. The four-square data were collected by [7] and were also analyzed in [4]. The other data are newly gathered. Each data set consists of approximately 4000 trials from each subject. To maintain their interest in the task, participants were given auditory feedback after each trial that indicated success or failure.

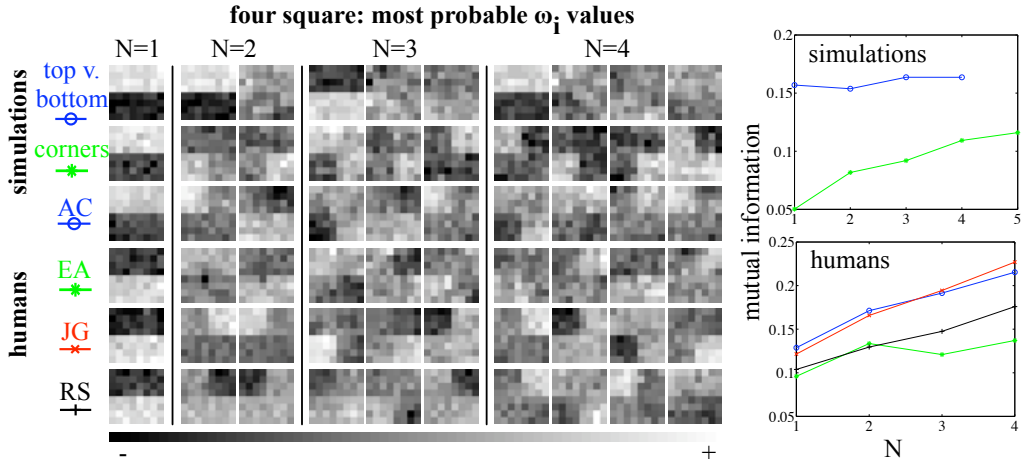

Figure 2: The most probable $\omega$ parameters found for the four-square experiments for different values of $N$ and the mutual information between these feature detectors and the observed classifications.

Fitting GRIFT models is not especially sensitive to the random initialization procedure used to start each EM instance. The $\lambda^*$ parameters were initialized by normal random samples and then half were negated so the features would tend to start evenly assigned to the two classes, except for $\lambda_0^*$, which was initialized to 0. In the four-square experiments, the $\omega^*$ parameters were initialized by a mixture of normal distributions and in the light-dark experiments they were initialized from a uniform distribution. In the faces experiments the $\omega^*$ were initialized by adding normal noise to the optimal linear classifier separating the two targets. Because of the large number of pixels in the faces stimuli, the other initialization procedures frequently produced initial assignments with extremely low probabilities, which led to numerical precision problems. In the four-square experiments, the $\beta^*$ were initialized randomly. In the other experiments, the intent was to set them to the optimal threshold for distinguishing the classes using the initial $\omega^*$ as a linear classifier, but a programming error set them to the negation of that value. In most cases, the results were insensitive to the choice of initialization method.

In the four-square experiment, the noise levels were continually adjusted to keep the participants' performance at approximately 71% using the *stair-casing* algorithm [8]. This performance level is high enough to keep the participants engaged in the task, but allows for sufficient noise to explore their responses in a large volume of the stimulus space. After an initial adaptation period, the noise level remains relatively constant across trials, so the inter-trial dependence introduced by the stair-casing can be safely ignored. Two simulated observers were created to validate GRIFT on the four-square task. Each used a GRIFT model with pre-specified parameters to probabilistically classify four-square data at a fixed noise level, which was chosen to produce approximately 70% correct performance. The *corners* observer used four feature detectors, one for each bright corner, whereas the *top-v.-bottom* observer contrasted the brightness of the top and bottom pixels.

The result of using GRIFT to recover the feature detectors are displayed in Figure 2. Only the $\omega$ parameters are displayed because they are the most informative. Dark pixels indicate negative weights and bright pixels correspond to positive weights. The presence of dark and light regions in a feature detector indicates the computation of contrasts between those areas. The sign of the weights is not significant — given a fixed number of features, there are typically several equivalent sets of feature detectors that only differ from each other in the signs of their $\omega$ terms and in the associated $\lambda$ and $\beta$ values.

Because the optimal number of features for human subjects is unknown, GRIFT models with 1–4 features were fit to the data from each subject. The correct number of features could be determined by holding out a test set or by performing cross-validation. Simulation demonstrated that a reliable test set would need to contain nearly all of the gathered samples, and computational expense made cross-validation impractical with our current MATLAB implementation. Instead, after recovering the parameters, we estimated the *mutual information* between the unobserved $F$ variables and the observed classifications $C$. Mutual information measures how well the feature detector outputs can

predict the subject's classification decision. Unlike the log likelihood of the observations, which is dependent on the choice to model $C$ with a logistic regression function, mutual information does not assume a particular relationship between $F$ and $C$ and does not necessarily increase with $N$. Plotting the mutual information as $N$ increases can indicate if new detectors are making a substantial contribution or are overfitting the data. On the simulated observers' data, for which the true values of $N$ were known, mutual information was a more accurate model selection indicator than traditional statistics such as the Bayesian or Akaike information criteria [3].

Fitting GRIFT to the simulated observers demonstrated that if the model is accurate, the correct features can be recovered reliably. The top-v.-bottom observer showed no substantial increase in mutual information as the number of features increased from 1 to 4. Each set of recovered feature detectors included a top-bottom contrast detector and other detectors with noisy $\omega_i$s that did not contribute much to predicting $C$. Although the observer truly used two detectors, one top-brighter detector and one bottom-brighter detector, the recovery of only one top-bottom contrast detector is a success because one contrast detector plus a suitable $\lambda_0$ term is logically equivalent to the original two-feature model. The corners observer showed a substantial increase in mutual information as $N$ increased from 1 to 4 and the $\omega$ values clearly indicate four corner-sensitive feature detectors. The corners data was also tested with a five-feature GRIFT model ($\omega$ not shown) which produced four corner detectors and one feature with noisy $\omega_i$. Its gain in mutual information was smaller than that observed on any of the previous steps. Note the corner areas in the $\omega_i$s recovered from the corners data are sometimes black and sometimes white. Recall that these are not image pixel values that the detectors are attempting to match, but positive and negative weights indicating that the brightness in the corner region is being contrasted to the brightness of the rest of the image.

Even though targets consisted of four bright-corner stimuli, recovering the parameters from the top-v.-bottom observer never produced $\omega$ values indicating corner-specific feature detectors. An important advantages of GRIFT over previous methods such as [4] is that targets will not "contaminate" the recovered detectors. The simulations demonstrate that the recovered detectors are determined by the classification strategy, not by the structure of the targets and classes.

The data of the four human participants revealed some interesting differences. Participants EA and RS were naive, while AC and JG were not. The largest disparity was between EA and JG. EA's data indicated no consistent pattern of mutual information increase after two features, and the two-feature model appears to contain two top-bottom contrast detectors. Therefore, it is reasonable to conclude that EA was not explicitly detecting the corners. At the other extreme is participant JG, whose data shows four very clear corner detectors and a steady increase in mutual information up to four features. Therefore, it seems very likely that this participant was matching corners and probably should be tested with a five-feature model to gain additional insight. AC and RS's data suggest three corner detectors and a top-bottom contrast detector. GRIFT's output indicates qualitative differences in the classification strategies used by the four human participants.

Across all participants, the best one-feature model was based on the contrast between the top of the image and the bottom. This is extremely similar to the result produced by a classification image of the data, reinforcing the strong similarity between one-feature GRIFT and that approach.

In the light-dark and faces experiments, stair-casing was used the adjust the noise level to the 71% performance level at the beginning of each session and then the noise level was fixed for the remaining trials to improve the independence of the samples. Participants were paid and promised a $10 reward for achieving the highest score on the task.

Participants P1, P2, and P3 classified the light-dark stimuli. P1 and P2 achieved at or above the expected performance level (82% and 73% accuracy), while P3's performance was near chance (55%). Because the noise levels were fixed after the first 101 trials, a participant with good luck at the end of that period could experience very high noise levels for the remainder of the experiment, leading to poor performance. All three participants appear to have used different classification methods, providing a very informative contrast. The results of fitting the GRIFT model are in Figure 3.

The flat mutual information graph and the presence of a feature detector thresholding the overall brightness for each value of $N$ indicate that P1 pursued a one-feature, linear-classifier strategy. P2, on the other hand, clearly employed a multi-feature, non-linear strategy. For $N = 1$ and $N = 2$, the most interpretable feature detector is an overall brightness detector, which disappears when $N = 3$ and the best fit model consists of three detectors looking for specific patterns, one for each position a

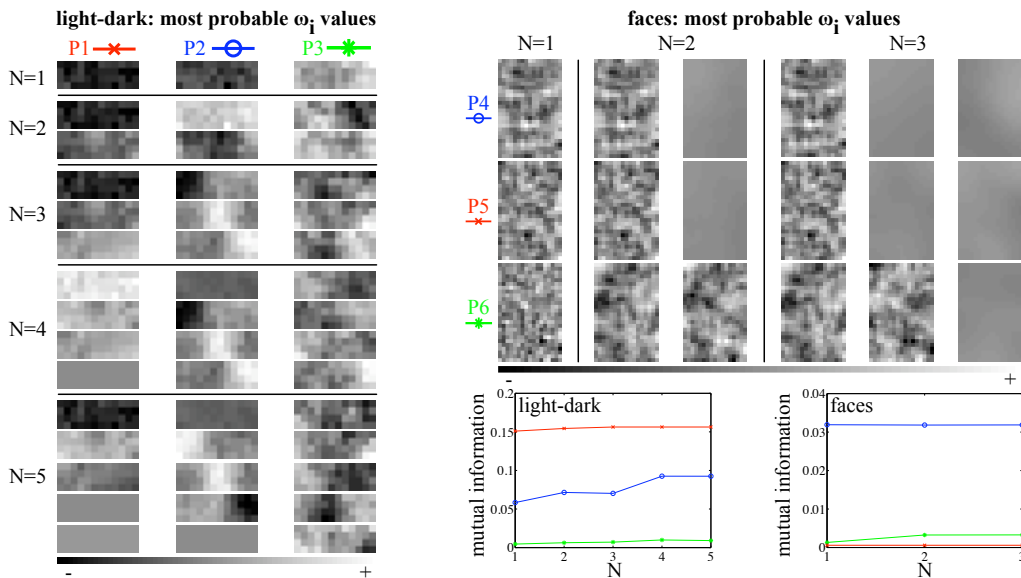

Figure 3: The most probable $\omega$ parameters found for the light-dark and faces experiments for different $N$ and the mutual information between these feature detectors and the observed classifications.

bright or dark spot can appear. Then when $N = 4$ the overall brightness detector reappears, added to the three spot detectors. Apparently the spot detectors are only effective if they are all present. With only three available detectors, the overall brightness detector is excluded, but the optimal assignment includes all four detectors. This is the best-fit model because increasing to $N = 5$ keeps the mutual information constant and adds a detector that is active for every stimulus. Always active detectors function as constant additions to $\lambda_0$, therefore this is equivalent to the $N = 4$ solution.

The GRIFT models of participant P3 do not show a substantial increase in mutual information as the number of features rises. This lack of increase leads to the conclusion that the one-feature model is probably the best fit, and since performance was extremely low, it can be assumed that the subject was reduced to near random guessing much of the time.

The clear distinction between the results for all three subjects demonstrates the effectiveness of GRIFT and the mutual information measure in distinguishing between classification strategies.

The faces presented the largest computational challenges. The targets were two unfiltered faces from Gold et al.'s data set [6], down-sampled to 128x128. After the experiment, the stimuli were down-sampled further to 32x32 and the background surrounding the faces was removed by cropping, reducing the stimuli to 26x17. These steps made the algorithm computationally feasible, and reduced the number of parameters so they would be sufficiently constrained by the samples.

The results for three participants (P4, P5, and P6) are in Figure 3. Participants P4 and P5's data were clearly best fit by one-feature GRIFT models. Increasing the number of features simply caused the algorithm to add features that were never or always active. Never active features cannot affect the classification, and, as explained previously, always active features are also superfluous. P4's one-feature model clearly places significant weight near the eyebrows, nose, and other facial features. P5's one-feature weights are much noisier and harder to interpret. This might be related to P5's poor performance on the task — only 53% accuracy compared to P4's 72% accuracy. Perhaps the noise level was too high and P5 was guessing rather than using image information much of the time.

Participant P6's data did produce a two-feature GRIFT model, albeit one that is difficult to interpret and which only caused a small rise in mutual information. Instead of recovering independent part detectors, such as a nose detector and an eye detector, GRIFT extracted two subtly different holistic feature detectors. Given P6's poor performance (58% accuracy) these features may, like P5's results, be indicative of a guessing strategy that was not strongly influenced by the image information.

The results on faces support the hypothesis that face classification is holistic and configural, rather than the result of part classifications, especially when individual feature detection is difficult [11].

Across these experiments, the data collected were compatible with the original classification image method. In fact, the four-square human data were originally analyzed using that algorithm. One of the advantages of GRIFT is that it can reanalyze old data to reveal new information. In the one-feature case, GRIFT enables the use of prior probabilities on the parameters, which may improve performance when data is too scarce for the classification image approach. Most importantly, fitting multi-feature GRIFT models can reveal previously hidden non-linear classification strategies.

## 6   Conclusion

This paper has described the GRIFT model for determining the features used in human image classification. GRIFT is an advance over previous methods that assume a single linear classifier on pixels because it describes classification as the combination of multiple independently detected features. It provides a probabilistic model of human visual classification that accounts for data and incorporates prior beliefs about the features. The feature detectors it finds are associated with the classification strategy employed by the observer and are not the result of structure in the classes' target images.

GRIFT's value has been demonstrated by modeling the performance of humans on the four-square, light-dark, and faces classification tasks and by successfully recovering the parameters of computer simulated observers in the four-square task. Its inability to find multiple local features when analyzing human performance on the faces task agrees with previous results.

One of the strengths of the graphical model approach is that it allows easy replacement of model components. An expert can easily change the prior distributions on the parameters to reflect knowledge gained in previous experiments. For example, it might be desirable to encourage the formation of edge detectors. New resolution-independent feature parameterizations can be introduced, as can transformation parameters to make the features translationally and rotationally invariant. If the features have explicitly parameterized locations and orientations, the model could be extended to model their joint relative positions, which might provide more information about domains such as face classification. The success of this version of GRIFT provides a firm foundation for these improvements.

**Acknowledgments**

This research was supported by NSF Grant SES-0631602 and NIMH grant MH16745. The authors thank the reviewers, Tom Griffiths, Erik Learned-Miller, and Adam Sanborn for their suggestions.

## References

[1] A.J. Ahumada, Jr. Classification image weights and internal noise level estimation. *Journal of Vision*, 2(1), 2002.

[2] C.M. Bishop. *Neural Networks for Pattern Recognition*. Oxford University Press, 1995.

[3] C.M. Bishop. *Pattern Recognition and Machine Learning*. Springer, 2006.

[4] A.L. Cohen, R.M. Shiffrin, J.M. Gold, D.A. Ross, and M.G. Ross. Inducing features from visual noise. *Journal of Vision*, 7(8), 2007.

[5] A. Gelman, J.B. Carlin, H.S. Stern, and D.B. Rubin. *Bayesian Data Analysis*. Chapman & Hall/CRC, 2003.

[6] J.M. Gold, P.J. Bennett, and A.B. Sekuler. Identification of band-pass filtered letters and faces by human and ideal observers. *Vision Research*, 39, 1999.

[7] J.M. Gold, A.L. Cohen, and R. Shiffrin. Visual noise reveals category representations. *Psychonomics Bulletin & Review*, 15(4), 2006.

[8] N.A. Macmillan and C.D. Creelman. *Detection Theory: A User's Guide*. Lawrence Erlbaum Associates, 2005.

[9] S.E. Palmer. *Vision Science: Photons to Phenomenology*. The MIT Press, 1999.

[10] D.G. Pelli, B. Farell, and D.C. Moore. The remarkable inefficiency of word recognition. *Nature*, 425, 2003.

[11] J. Sergent. An investigation into component and configural processes underlying face perception. *British Journal of Psychology*, 75, 1984.

